# Clustering via Concave Minimization

**P. S. Bradley and O. L. Mangasarian**
Computer Sciences Department
University of Wisconsin
1210 West Dayton Street
Madison, WI 53706
email: *paulb@cs.wisc.edu, olvi@cs.wisc.edu*

**W. N. Street**
Computer Science Department
Oklahoma State University
205 Mathematical Sciences
Stillwater, OK 74078
email:*nstreet@cs.okstate.edu*

## Abstract

The problem of assigning $m$ points in the $n$-dimensional real space $R^n$ to $k$ clusters is formulated as that of determining $k$ centers in $R^n$ such that the sum of distances of each point to the nearest center is minimized. If a polyhedral distance is used, the problem can be formulated as that of minimizing a piecewise-linear concave function on a polyhedral set which is shown to be equivalent to a bilinear program: minimizing a bilinear function on a polyhedral set. A fast finite $k$-Median Algorithm consisting of solving few linear programs in closed form leads to a stationary point of the bilinear program. Computational testing on a number of real-world databases was carried out. On the Wisconsin Diagnostic Breast Cancer (WDBC) database, $k$-Median training set correctness was comparable to that of the $k$-Mean Algorithm, however its testing set correctness was better. Additionally, on the Wisconsin Prognostic Breast Cancer (WPBC) database, distinct and clinically important survival curves were extracted by the $k$-Median Algorithm, whereas the $k$-Mean Algorithm failed to obtain such distinct survival curves for the same database.

## 1 Introduction

The unsupervised assignment of elements of a given set to groups or clusters of like points, is the objective of cluster analysis. There are many approaches to this problem, including statistical [9], machine learning [7], integer and mathematical programming [18, 1]. In this paper we concentrate on a simple concave minimization formulation of the problem that leads to a finite and fast algorithm. Our point of

departure is the following explicit description of the problem: given $m$ points in the $n$-dimensional real space $R^n$, and a fixed number $k$ of clusters, determine $k$ centers in $R^n$ such that the sum of "distances" of each point to the nearest center is minimized. If the 1-norm is used, the problem can be formulated as the minimization of a piecewise-linear concave function on a polyhedral set. This is a hard problem to solve because a local minimum is not necessarily a global minimum. However, by converting this problem to a bilinear program, a fast successive-linearization $k$-Median Algorithm terminates after a few linear programs (each explicitly solvable in closed form) at a point satisfying the minimum principle necessary optimality condition for the problem. Although there is no guarantee that such a point is a global solution to our original problem, numerical tests on five real-world databases indicate that the $k$-Median Algorithm is comparable to or better than the $k$-Mean Algorithm [18, 9, 8]. This may be due to the fact that outliers have less influence on the $k$-Median Algorithm which utilizes the 1-norm distance. In contrast the $k$-Mean Algorithm uses squares of 2-norm distances to generate cluster centers which may be inaccurate if outliers are present. We also note that clustering algorithms based on statistical assumptions that minimize some function of scatter matrices do not appear to have convergence proofs [8, pp. 508-515], however convergence to a partial optimal solution is given in [18] for $k$-Mean type algorithms.

We outline now the contents of the paper. In Section 2, we formulate the clustering problem for a fixed number of clusters, as that of minimizing the sum of the 1-norm distances of each point to the nearest cluster center. This piecewise-linear concave function minimization on a polyhedral set turns out to be equivalent to a bilinear program [3]. We use an effective linearization of the bilinear program proposed in [3, Algorithm 2.1] to solve our problem by solving a few linear programs. Because of the simple structure, these linear programs can be explicitly solved in closed form, thus leading to the finite $k$-Median Algorithm 2.3 below. In Section 3 we give computational results on five real-world databases. Section 4 concludes the paper.

A word about our notation now. All vectors are column vectors unless otherwise specified. For a vector $x \in R^n$, $x_i$, $i = 1, \ldots, n$, will denote its components. The norm $\| \cdot \|_p$ will denote the $p$ norm, $1 \le p \le \infty$, while $A \in R^{m \times n}$ will signify a real $m \times n$ matrix. For such a matrix, $A^T$ will denote the transpose, and $A_i$ will denote row $i$. A vector of ones in a real space of arbitrary dimension will be denoted by $e$.

## 2    Clustering as Bilinear Programming

Given a set $\mathcal{A}$ of $m$ points in $R^n$ represented by the matrix $A \in R^{m \times n}$ and a number $k$ of desired clusters, we formulate the clustering problem as follows. Find cluster centers $C_\ell$, $\ell = 1, \ldots, k$, in $R^n$ such that the sum of the minima over $\ell \in \{1, \ldots, k\}$ of the 1-norm distance between each point $A_i$, $i = 1, \ldots, m$, and the cluster centers $C_\ell$, $\ell = 1, \ldots, k$, is minimized. More specifically we need to solve the following mathematical program:

$$\underset{C,D}{\text{minimize}} \qquad \sum_{i=1}^{m} \min_{\ell=1,\ldots,k} \{e^T D_{i\ell}\} \tag{1}$$
$$\text{subject to} \quad -D_{i\ell} \le A_i^T - C_\ell \le D_{i\ell}, \; i = 1, \ldots, m, \; \ell = 1, \ldots k$$

Here $D_{i\ell} \in R^n$, is a dummy variable that bounds the components of the difference

$A_i^T - C_\ell$ between point $A_i^T$ and center $C_\ell$, and $e$ is a vector of ones in $R^n$. Hence $e^T D_{i\ell}$ bounds the 1-norm distance between $A_i$ and $C_\ell$. We note immediately that since the objective function of (1) is the sum of minima of $k$ linear (and hence concave) functions, it is a piecewise-linear concave function [13, Corollary 4.1.14]. If the 2-norm or $p$-norm, $p \neq 1, \infty$, is used, the objective function will be neither concave nor convex. Nevertheless, minimizing a piecewise-linear concave function on a polyhedral set is NP-hard, because the general linear complementarity problem, which is NP-complete [4], can be reduced to such a problem [11, Lemma 1]. Given this fact we try to look for effective methods for processing this problem. We propose reformulation of problem (1) as a bilinear program. Such reformulations have been very effective in computationally solving NP-complete linear complementarity problems [14] as well as other difficult machine learning [12] and optimization problems with equilibrium constraints [12]. In order to carry out this reformulation we need the following simple lemma.

**Lemma 2.1** *Let $a \in R^k$. Then*

$$\min_{1 \le \ell \le k} \{a_\ell\} = \min_{t \in R^k} \left\{ \sum_{\ell=1}^k a_\ell t_\ell \;\middle|\; \sum_{\ell=1}^k t_\ell = 1, \, t_\ell \ge 0, \, \ell = 1, \ldots, k \right\} \qquad (2)$$

**Proof** This essentially obvious result follows immediately upon writing the dual of the linear program appearing on the right-hand side of (2) which is

$$\max_{h \in R} \{h \mid h \le a_\ell, \, \ell = 1, \ldots k\} \qquad (3)$$

Obviously, the maximum of this dual problem is $h = \min_{1 \le \ell \le k} \{a_\ell\}$. By linear programming duality theory, this maximum equals the minimum of the primal linear program in the right hand side of (2). This establishes the equality of (2). □

By defining $a_\ell^i = e^T D_{i\ell}$, $i = 1, \ldots, m$, $\ell = 1, \ldots, k$, Lemma 2.1 can be used to reformulate the clustering problem (1) as a bilinear program as follows.

**Proposition 2.2 Clustering as a Bilinear Program** *The clustering problem (1) is equivalent to the following bilinear program:*

$$\underset{C_\ell \in R^n, D_{i\ell} \in R^n, T_{i\ell} \in R}{\text{minimize}} \qquad \sum_{i=1}^m \sum_{\ell=1}^k e^T D_{i\ell} T_{i\ell}$$
$$\text{subject to} \qquad -D_{i\ell} \le A_i^T - C_\ell \le D_{i\ell}, i = 1 \ldots, m, \, \ell = 1, \ldots, k \qquad (4)$$
$$\sum_{\ell=1}^k T_{i\ell} = 1 \qquad T_{i\ell} \ge 0, \, i = 1, \ldots, m, \, \ell = 1, \ldots, k$$

Note that the constraints of (4) are uncoupled in the variables $(C, D)$ and the variable $T$. Hence the Uncoupled Bilinear Program Algorithm UBPA [3, Algorithm 2.1] is applicable. Simply stated, this algorithm alternates between solving a linear program in the variable $T$ and a linear program in the variables $(C, D)$. The algorithm terminates in a finite number of iterations at a stationary point satisfying the minimum principle necessary optimality condition for problem (4) [3, Theorem 2.1]. We note however, because of the simple structure the bilinear program (4), the two linear programs can be solved explicitly in closed form. This leads to the following algorithmic implementation.

**Algorithm 2.3 k-Median Algorithm** *Given $C_1^j, \ldots, C_k^j$ at iteration $j$, compute $C_1^{j+1}, \ldots, C_k^{j+1}$ by the following two steps:*

(a) **Cluster Assignment:** *For each $A_i^T$, $i = 1, \ldots m$, determine $\ell(i)$ such that $C_{\ell(i)}^j$ is closest to $A_i^T$ in the 1-norm.*

(b) **Cluster Center Update:** *For $\ell = 1, \ldots, k$ choose $C_\ell^{j+1}$ as a median of all $A_i^T$ assigned to $C_\ell^j$.*

*Stop when $C_\ell^{j+1} = C_\ell^j$, $\ell = 1, \ldots, k$.*

Although the $k$-Median Algorithm is similar to the $k$-Mean Algorithm wherein the 2-norm distance is used [18, 8, 9], it differs from it computationally, and theoretically. In fact, the underlying problem (1) of the $k$-Median Algorithm is a concave minimization on a polyhedral set while the corresponding problem for the $p$-norm, $p \neq 1$, is:

$$
\begin{aligned}
\underset{C,D}{\text{minimize}} \quad & \sum_{i=1}^{m} \min_{\ell=1,\ldots,k} \|D_{i\ell}\|_p \\
\text{subject to} \quad & -D_{i\ell} \leq A_i^T - C_\ell \leq D_{i\ell}, i = 1 \ldots, m, \; \ell = 1, \ldots, k.
\end{aligned}
\tag{5}
$$

This is not a concave minimization on a polyhedral set, because the minimum of a set of convex functions is not in general concave. The concave minimization problem of [18] is not in the original space of the problem variables, that is, the cluster center variables, $(C, D)$, but merely in the space of variables $T$ that assign points to clusters. We also note that the $k$-Mean Algorithm finds a stationary point not of problem (5) with $p = 2$, but of the same problem except that $\|D_{i\ell}\|_2$ is replaced by $\|D_{i\ell}\|_2^2$. Without this squared distance term, the subproblem of the $k$-Mean Algorithm becomes the considerably harder Weber problem [17, 5] which locates a center in $R^n$ closest in sum of Euclidean distances (not their squares!) to a finite set of given points. The Weber problem has no closed form solution. However, using the mean as a cluster center of points assigned to the cluster, minimizes the sum of the *squares* of the distances from the cluster center to the points. It is precisely the mean that is used in the $k$-Mean Algorithm subproblem.

Because there is no guaranteed way to ensure global optimality of the solution obtained by either the $k$-Median or $k$-Mean Algorithms, different starting points can be used to initiate the algorithm. Random starting cluster centers or some other heuristic can be used such as placing $k$ initial centers along the coordinate axes at densest, second densest, ..., $k$ densest intervals on the axes.

## 3   Computational Results

An important computational issue is how to measure the correctness of the results obtained by the proposed algorithm. We decided on the following three ways.

**Remark 3.1 Training Set Correctness** *The $k$-Median algorithm $(k = 2)$ is applied to a database with two known classes to obtain centers. Training correctness is measured by the ratio of the sum of the number examples of the majority class in each cluster to the total number of points in the database. The $k$-Median training set correctness is compared to that of the $k$-Mean Algorithm as well as the training correctness of a supervised learning method, a perceptron trained by robust linear programming [2]. Table 1 shows results averaged over ten random starts for the*

*publicly available Wisconsin Diagnostic Breast Cancer (WDBC) database as well as three others [15, 16]. We note that for two of the databases k-Median outperformed k-Mean, and for the other two k-Mean was better.*

| Algorithm ↓ Database → | WDBC | Cleveland | Votes | Star/Galaxy-Bright |
|---|---|---|---|---|
| Unsupervised *k*-Median | 93.2% | 80.6% | 84.6% | 87.6% |
| Unsupervised *k*-Mean | 91.1% | 83.1% | 85.5% | 85.6% |
| Supervised Robust LP | 100% | 86.5% | 95.6% | 99.7% |

**Table 1** Training set correctness using the unsupervised *k*-Median
and *k*-Mean Algorithms and the supervised Robust LP on four databases

**Remark 3.2 Testing Set Correctness**

*The idea behind this approach is that supervised learning may be costly due to problem size, difficulty in obtaining true classification, etc., hence the importance of good performance of an unsupervised learning algorithm on a testing subset of a database. The WDBC database [15] is split into training and testing subsets of different proportions. The k-Median and k-Mean Algorithms (k = 2) are applied to the training subset. The centers are given class labels determined by the majority class of training subset points assigned to the cluster. Class labels are assigned to the testing subset by the label of the clos-*

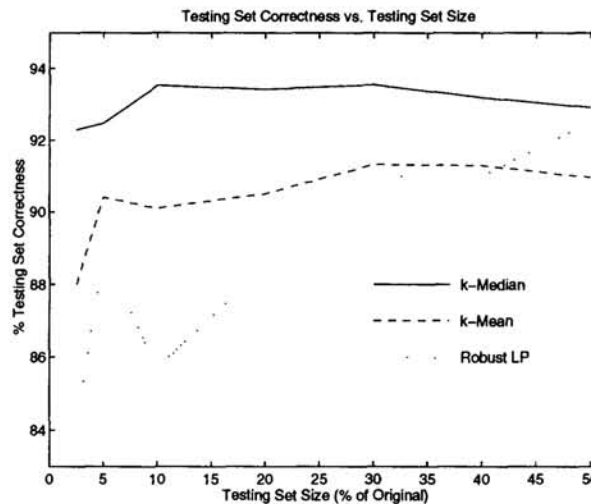

Figure 1: Correctness on variable-size test set of unsupervised *k*-Median & *k*-Mean Algorithms versus correctness of the supervised Robust LP on WDBC

*est center. Testing correctness is determined by the number of points in testing subset correctly classified by this assignment. This is compared to the correctness of a supervised learning method, a perceptron trained via robust linear programming [2], using the leave-one-out strategy applied to the testing subset only. This comparison is then carried out for various sizes of the testing subset. Figure 1 shows the results averaged over 50 runs for each of 7 testing subset sizes. As expected, the performance of the supervised learning algorithm (Robust LP) improved as the size of the testing subset increases. The k-Median Algorithm test set correctness remained fairly constant in the range of 92.3% to 93.5%, while the k-Mean Algorithm test set correctness was lower and more varied in the range 88.0% to 91.3%.*

**Remark 3.3 Separability of Survival Curves**   *In mining medical databases, survival curves [10] are important prognostic tools. We applied the k-Median and k-Mean (k = 3) Algorithms, as knowledge discovery in database (KDD) tools [6], to the Wisconsin Prognostic Breast Cancer Database (WPBC) [15] using only two features: tumor size and lymph node status. Survival curves were constructed for*

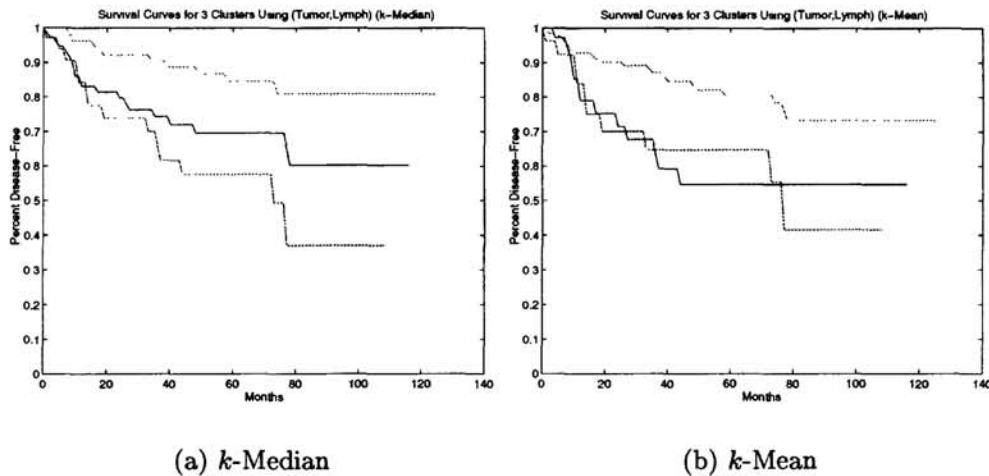

(a) $k$-Median             (b) $k$-Mean

Figure 2: Survival curves for the 3 clusters obtained by $k$-Median and $k$-Mean Algorithms

*each cluster, representing expected percent of surviving patients as a function of time, for patients in that cluster. Figure 2(a) depicts the survival curves from clusters obtained from the k-Median Algorithm, Figure 2(b) depicts curves for the k-Mean Algorithm. The key observation to make here is that curves in Figure 2(a) are well separated, and hence the clusters can be used as prognostic indicators. In contrast, the curves in Figure 2(b) are poorly separated, and hence are not useful for prognosis.*

## 4   Conclusion

We have proposed a new approach for assigning points to clusters based on a simple concave minimization model. Although a global solution to the problem cannot be guaranteed, a finite and simple $k$-Median Algorithm quickly locates a very useful stationary point. Utility of the proposed algorithm lies in its ability to handle large databases and hence would be a useful tool for data mining. Comparing it with the $k$-Mean Algorithm, we have exhibited instances where the $k$-Median Algorithm is superior, and hence preferable. Further research is needed to pinpoint types of problems for which the $k$-Median Algorithm is best.

## 5   Acknowledgements

Our colleague Jude Shavlik suggested the testing set strategy used in Remark 3.2. This research is supported by National Science Foundation Grants CCR-9322479 and National Institutes of Health INRSA Fellowship 1 F32 CA 68690-01.

## References

[1] K. Al-Sultan. A Tabu search approach to the clustering problem. *Pattern Recognition*, 28(9):1443–1451, 1995.

[2] K. P. Bennett and O. L. Mangasarian. Robust linear programming discrimination of two linearly inseparable sets. *Optimization Methods and Software*, 1:23–34, 1992.

[3] K. P. Bennett and O. L. Mangasarian. Bilinear separation of two sets in n-space. *Computational Optimization & Applications*, 2:207–227, 1993.

[4] S.-J. Chung. NP-completeness of the linear complementarity problem. *Journal of Optimization Theory and Applications*, 60:393–399, 1989.

[5] F. Cordellier and J. Ch. Fiorot. On the Fermat-Weber problem with convex cost functionals. *Mathematical Programming*, 14:295–311, 1978.

[6] U. Fayyad, G. Piatetsky-Shapiro, and P. Smyth. The KDD process for extracting useful knowledge from volumes of data. *Communications of the ACM*, 39:27–34, 1996.

[7] D. Fisher. Knowledge acquisition via incremental conceptual clustering. *Machine Learning*, 2:139–172, 1987.

[8] K. Fukunaga. *Statistical Pattern Recognition*. Academic Press, NY, 1990.

[9] A. K. Jain and R. C. Dubes. *Algorithms for Clustering Data*. Prentice-Hall, Inc, Englewood Cliffs, NJ, 1988.

[10] E. L. Kaplan and P. Meier. Nonparametric estimation from incomplete observations. *J. Am. Stat. Assoc.*, 53:457–481, 1958.

[11] O. L. Mangasarian. Characterization of linear complementarity problems as linear programs. *Mathematical Programming Study*, 7:74–87, 1978.

[12] O. L. Mangasarian. Misclassification minimization. *Journal of Global Optimization*, 5:309–323, 1994.

[13] O. L. Mangasarian. *Nonlinear Programming*. SIAM, Philadelphia, PA, 1994.

[14] O. L. Mangasarian. The linear complementarity problem as a separable bilinear program. *Journal of Global Optimization*, 6:153–161, 1995.

[15] P. M. Murphy and D. W. Aha. UCI repository of machine learning databases. Department of Information and Computer Science, University of California, Irvine, www.ics.uci.edu/AI/ML/MLDBRepository.html, 1992.

[16] S. Odewahn, E. Stockwell, R. Pennington, R. Hummphreys, and W. Zumach. Automated star/galaxy discrimination with neural networks. *Astronomical Journal*, 103(1):318–331, 1992.

[17] M. L. Overton. A quadratically convergent method for minimizing a sum of euclidean norms. *Mathematical Programming*, 27:34–63, 1983.

[18] S. Z. Selim and M. A. Ismail. K-Means-Type algorithms: a generalized convergence theorem and characterization of local optimality. *IEEE Transactions on Pattern Analysis and Machine Intelligence*, PAMI-6:81–87, 1984.